# Improved Deep Metric Learning with Multi-class N-pair Loss Objective

**Kihyuk Sohn**
NEC Laboratories America, Inc.
`ksohn@nec-labs.com`

## Abstract

Deep metric learning has gained much popularity in recent years, following the success of deep learning. However, existing frameworks of deep metric learning based on contrastive loss and triplet loss often suffer from slow convergence, partially because they employ only one negative example while not interacting with the other negative classes in each update. In this paper, we propose to address this problem with a new metric learning objective called *multi-class N-pair loss*. The proposed objective function firstly generalizes triplet loss by allowing joint comparison among more than one negative examples – more specifically, $N$-1 negative examples – and secondly reduces the computational burden of evaluating deep embedding vectors via an efficient batch construction strategy using only $N$ pairs of examples, instead of $(N+1)\times N$. We demonstrate the superiority of our proposed loss to the triplet loss as well as other competing loss functions for a variety of tasks on several visual recognition benchmark, including fine-grained object recognition and verification, image clustering and retrieval, and face verification and identification.

## 1 Introduction

Distance metric learning aims to learn an embedding representation of the data that preserves the distance between similar data points close and dissimilar data points far on the embedding space [15, 30]. With success of deep learning [13, 20, 23, 5], deep metric learning has received a lot of attention. Compared to standard distance metric learning, it learns a nonlinear embedding of the data using deep neural networks, and it has shown a significant benefit by learning deep representation using contrastive loss [3, 7] or triplet loss [27, 2] for applications such as face recognition [24, 22, 19] and image retrieval [26]. Although yielding promising progress, such frameworks often suffer from slow convergence and poor local optima, partially due to that the loss function employs only one negative example while not interacting with the other negative classes per each update. Hard negative data mining could alleviate the problem, but it is expensive to evaluate embedding vectors in deep learning framework during hard negative example search. As to experimental results, only a few has reported strong empirical performance using these loss functions alone [19, 26], but many have combined with softmax loss to train deep networks [22, 31, 18, 14, 32].

To address this problem, we propose an $(N+1)$-tuplet loss that optimizes to identify a positive example from $N$-1 negative examples. Our proposed loss extends triplet loss by allowing joint comparison among more than one negative examples; when $N$=2, it is equivalent to triplet loss. One immediate concern with $(N+1)$-tuplet loss is that it quickly becomes intractable when scaling up since the number of examples to evaluate in each batch grows in quadratic to the number of tuplets and their length $N$. To overcome this, we propose an efficient batch construction method that only requires $2N$ examples instead of $(N+1)N$ to build $N$ tuplets of length $N+1$. We unify the $(N+1)$-tuplet loss with our proposed batch construction method to form a novel, scalable and effective deep metric learning objective, called *multi-class N-pair loss (N-pair-mc loss)*. Since the $N$-pair-mc

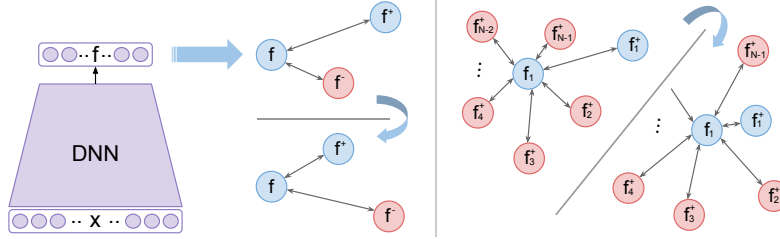

Figure 1: Deep metric learning with (left) triplet loss and (right) $(N+1)$-tuplet loss. Embedding vectors $f$ of deep networks are trained to satisfy the constraints of each loss. Triplet loss pulls positive example while pushing one negative example at a time. On the other hand, $(N+1)$-tuplet loss pushes $N$-1 negative examples *all at once*, based on their similarity to the input example.

loss already considers comparison to $N$-1 negative examples in its training objectives, negative data mining won't be necessary in learning from small or medium-scale datasets in terms of the number of output classes. For datasets with large number of output classes, we propose a hard negative "class" mining scheme which greedily adds examples to form a batch from a class that violates the constraint with the previously selected classes in the batch.

In experiment, we demonstrate the superiority of our proposed $N$-pair-mc loss to the triplet loss as well as other competing metric learning objectives on visual recognition, verification, and retrieval tasks. Specifically, we report much improved recognition and verification performance on our fine-grained car and flower recognition datasets. In comparison to the softmax loss, $N$-pair-mc loss is as competitive for recognition but significantly better for verification. Moreover, we demonstrate substantial improvement in image clustering and retrieval tasks on Online product [21], Car-196 [12], and CUB-200 [25], as well as face verification and identification accuracy on LFW database [8].

## 2 Preliminary: Distance Metric Learning

Let $x \in \mathcal{X}$ be an input data and $y \in \{1, \cdots, L\}$ be its output label. We use $x^+$ and $x^-$ to denote positive and negative examples of $x$, meaning that $x$ and $x^+$ are from the same class and $x^-$ is from different class to $x$. The kernel $f(\cdot; \theta) : \mathcal{X} \to \mathbb{R}^K$ takes $x$ and generates an embedding vector $f(x)$. We often omit $x$ from $f(x)$ for simplicity, while $f$ inherits all superscripts and subscripts.

Contrastive loss [3, 7] takes pairs of examples as input and trains a network to predict whether two inputs are from the same class or not. Specifically, the loss is written as follows:

$$\mathcal{L}_{\text{cont}}^m(x_i, x_j; f) = \mathbf{1}\{y_i = y_j\}\|f_i - f_j\|_2^2 + \mathbf{1}\{y_i \neq y_j\}\max\left(0, m - \|f_i - f_j\|_2\right)^2 \quad (1)$$

where $m$ is a margin parameter imposing the distance between examples from different classes to be larger than $m$. Triplet loss [27, 2, 19] shares a similar spirit to contrastive loss, but is composed of triplets, each consisting of a query, a positive example (to the query), and a negative example:

$$\mathcal{L}_{\text{tri}}^m(x, x^+, x^-; f) = \max\left(0, \|f - f^+\|_2^2 - \|f - f^-\|_2^2 + m\right) \quad (2)$$

Compared to contrastive loss, triplet loss only requires the difference of (dis-)similarities between positive and negative examples to the query point to be larger than a margin $m$. Despite their wide use, both loss functions are known to suffer from slow convergence and they often require expensive data sampling method to provide nontrivial pairs or triplets to accelerate the training [2, 19, 17, 4].

## 3 Deep Metric Learning with Multiple Negative Examples

The fundamental philosophy behind triplet loss is the following: for an input (query) example, we desire to shorten the distances between its embedding vector and those of positive examples while enlarging the distances between that of negative examples. However, during one update, the triplet loss only compares an example with one negative example while ignoring negative examples from the rest of the classes. As a consequence, the embedding vector for an example is only guaranteed to be far from the selected negative class but not necessarily the others. Thus we can end up only differentiating an example from a limited selection of negative classes yet still maintain a small distance from many other classes. In practice, the hope is that, after looping over sufficiently many randomly sampled triplets, the final distance metric can be balanced correctly; but individual update can still be unstable and the convergence would be slow. Specifically, towards the end of training, most randomly selected negative examples can no longer yield non-zero triplet loss error.

An evident way to improve the vanilla triplet loss is to select a negative example that violates the triplet constraint. However, hard negative data mining can be expensive with a large number of output classes for deep metric learning. We seek an alternative: a loss function that recruits multiple negatives for each update, as illustrated by Figure 1. In this case, an input example is being compared against negative examples from multiple classes and it needs to be distinguishable from all of them at the same time. Ideally, we would like the loss function to incorporate examples across every class all at once. But it is usually not attainable for large scale deep metric learning due to the memory bottleneck from the neural network based embedding. Motivated by this thought process, we propose a novel, computationally feasible loss function, illustrated by Figure 2, which approximates our ideal loss by pushing $N$ examples simultaneously.

## 3.1 Learning to identify from multiple negative examples

We formalize our proposed method, which is optimized to identify a positive example from multiple negative examples. Consider an $(N{+}1)$-tuplet of training examples $\{x, x^+, x_1, \cdots, x_{N-1}\}$: $x^+$ is a positive example to $x$ and $\{x_i\}_{i=1}^{N-1}$ are negative. The $(N{+}1)$-tuplet loss is defined as follows:

$$\mathcal{L}(\{x, x^+, \{x_i\}_{i=1}^{N-1}\}; f) = \log\left(1 + \sum_{i=1}^{N-1} \exp(f^\top f_i - f^\top f^+)\right) \quad (3)$$

where $f(\cdot; \theta)$ is an embedding kernel defined by deep neural network. Recall that it is desirable for the tuplet loss to involve negative examples across all classes but it is impractical in the case when the number of output classes $L$ is large; even if we restrict the number of negative examples per class to one, it is still too heavy-lifting to perform standard optimization, such as stochastic gradient descent (SGD), with a mini-batch size as large as $L$.

When $N = 2$, the corresponding $(2{+}1)$-tuplet loss highly resembles the triplet loss as there is only one negative example for each pair of input and positive examples:

$$\mathcal{L}_{(2+1)\text{-tuplet}}(\{x, x^+, x_i\}; f) = \log\left(1 + \exp(f^\top f_i - f^\top f^+)\right); \quad (4)$$

$$\mathcal{L}_{\text{triplet}}(\{x, x^+, x_i\}; f) = \max\left(0, f^\top f_i - f^\top f^+\right). \quad (5)$$

Indeed, under mild assumptions, we can show that an embedding $f$ minimizes $\mathcal{L}_{(2+1)\text{-tuplet}}$ if and only if it minimizes $\mathcal{L}_{\text{triplet}}$, i.e., two loss functions are equivalent.[1] When $N > 2$, we further argue the advantages of $(N{+}1)$-tuplet loss over triplet loss. We compare $(N{+}1)$-tuplet loss with the triplet loss in terms of partition function estimation of an ideal $(L{+}1)$-tuplet loss, where an $(L{+}1)$-tuplet loss coupled with a single example per negative class can be written as follows:

$$\log\left(1 + \sum_{i=1}^{L-1} \exp(f^\top f_i - f^\top f^+)\right) = -\log\frac{\exp(f^\top f^+)}{\exp(f^\top f^+) + \sum_{i=1}^{L-1}\exp(f^\top f_i)} \quad (6)$$

Equation (6) is similar to the multi-class logistic loss (i.e., softmax loss) formulation when we view $f$ as a feature vector, $f^+$ and $f_i$'s as weight vectors, and the denominator on the right hand side of Equation (6) as a partition function of the likelihood $P(y = y^+)$. We observe that the partition function corresponding to the $(N{+}1)$-tuplet approximates that of $(L{+}1)$-tuplet, and larger the value of $N$, more accurate the approximation. Therefore, it naturally follows that $(N{+}1)$-tuplet loss is a better approximation than the triplet loss to an ideal $(L{+}1)$-tuplet loss.

## 3.2 $N$-pair loss for efficient deep metric learning

Suppose we directly apply the $(N{+}1)$-tuplet loss to the deep metric learning framework. When the batch size of SGD is $M$, there are $M{\times}(N{+}1)$ examples to be passed through $f$ at one update. Since the number of examples to evaluate for each batch grows in quadratic to $M$ and $N$, it again becomes impractical to scale the training for a very deep convolutional networks.

Now, we introduce an effective batch construction to avoid excessive computational burden. Let $\{(x_1, x_1^+), \cdots, (x_N, x_N^+)\}$ be $N$ pairs of examples from $N$ different classes, i.e., $y_i \neq y_j, \forall i \neq j$. We build $N$ tuplets, denoted as $\{S_i\}_{i=1}^N$, from the $N$ pairs, where $S_i = \{x_i, x_1^+, x_2^+, \cdots, x_N^+\}$. Here, $x_i$ is the query for $S_i$, $x_i^+$ is the positive example and $x_j^+, j \neq i$ are the negative examples.

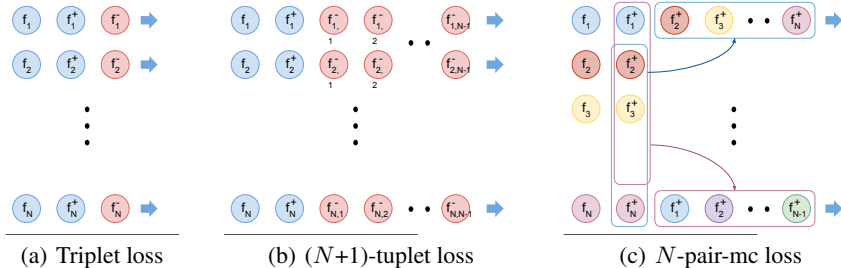

(a) Triplet loss    (b) $(N{+}1)$-tuplet loss    (c) $N$-pair-mc loss

Figure 2: Triplet loss, $(N{+}1)$-tuplet loss, and multi-class $N$-pair loss with training batch construction. Assuming each pair belongs to a different class, the $N$-pair batch construction in (c) leverages all $2 \times N$ embedding vectors to build $N$ distinct $(N{+}1)$-tuplets with $\{f_i\}_{i=1}^N$ as their queries; thereafter, we congregate these $N$ distinct tuplets to form the $N$-pair-mc loss. For a batch consisting of $N$ distinct queries, triplet loss requires $3N$ passes to evaluate the necessary embedding vectors, $(N{+}1)$-tuplet loss requires $(N{+}1)N$ passes and our $N$-pair-mc loss only requires $2N$.

Figure 2(c) illustrates this batch construction process. The corresponding $(N{+}1)$-tuplet loss, which we refer to as the *multi-class N-pair loss (N-pair-mc)*, can be formulated as follows:[2]

$$\mathcal{L}_{N\text{-pair-mc}}(\{(x_i, x_i^+)\}_{i=1}^N; f) = \frac{1}{N}\sum_{i=1}^N \log\left(1 + \sum_{j \neq i} \exp(f_i^\top f_j^+ - f_i^\top f_i^+)\right) \tag{7}$$

The mathematical formulation of our $N$-pair loss shares similar spirits with other existing methods, such as the neighbourhood component analysis (NCA) [6] and the triplet loss with lifted structure [21].[3] Nevertheless, our batch construction is designed to achieve the utmost potential of such $(N{+}1)$-tuplet loss, when using deep CNNs as embedding kernel on large scale datasets both in terms of training data and number of output classes. Therefore, the proposed $N$-pair-mc loss is a novel framework consisting of two indispensable components: the $(N{+}1)$-tuplet loss, as the building block loss function, and the $N$-pair construction, as the key to enable highly scalable training. Later in Section 4.4, we empirically show the advantage of our $N$-pair-mc loss framework in comparison to other variations of mini-batch construction methods.

Finally, we note that the tuplet batch construction is not specific to the $(N{+}1)$-tuplet loss. We call the set of loss functions using tuplet construction method an *N-pair loss*. For example, when integrated into the standard triplet loss, we obtain the following *one-vs-one N-pair loss (N-pair-ovo)*:

$$\mathcal{L}_{N\text{-pair-ovo}}(\{(x_i, x_i^+)\}_{i=1}^N; f) = \frac{1}{N}\sum_{i=1}^N \sum_{j \neq i} \log\left(1 + \exp(f_i^\top f_j^+ - f_i^\top f_i^+)\right). \tag{8}$$

### 3.2.1 Hard negative class mining

The hard negative data mining is considered as an essential component to many triplet-based distance metric learning algorithms [19, 17, 4] to improve convergence speed or the final discriminative performance. When the number of output classes are not too large, it may be unnecessary for $N$-pair loss since the examples from most of the negative classes are considered jointly already. When we train on the dataset with large output classes, the $N$-pair loss can be benefited from carefully selected impostor examples.

Evaluating deep embedding vectors for multiple examples from large number of classes is computationally demanding. Moreover, for $N$-pair loss, one theoretically needs $N$ classes that are negative to one another, which substantially adds to the challenge of hard negative search. To overcome such difficulty, we propose negative "class" mining, as opposed to negative "instance" mining, which greedily selects negative classes in a relatively efficient manner.

More specifically, the negative class mining for $N$-pair loss can be executed as follows:

1. **Evaluate Embedding Vectors:** choose randomly a large number of output classes $C$; for each class, randomly pass a few (one or two) examples to extract their embedding vectors.

2. **Select Negative Classes:** select one class randomly from $C$ classes from step 1. Next, greedily add a new class that violates triplet constraint the most w.r.t. the selected classes till we reach $N$ classes. When a tie appears, we randomly pick one of tied classes [28].

3. **Finalize $N$-pair:** draw two examples from each selected class from step 2.

### 3.2.2   $L^2$ norm regularization of embedding vectors

The numerical value of $f^\top f^+$ can be influenced by not only the direction of $f^+$ but also its norm, even though the classification decision should be determined merely by the direction. Normalization can be a solution to avoid such situation, but it is too stringent for our loss formulation since it bounds the value of $|f^\top f^+|$ to be less than 1 and makes the optimization difficult. Instead, we regularize the $L^2$ norm of the embedding vectors to be small.

## 4   Experimental Results

We assess the impact of our proposed $N$-pair loss functions, such as multi-class $N$-pair loss ($N$-pair-mc) or one-vs-one $N$-pair loss ($N$-pair-ovo), on several generic and fine-grained visual recognition and verification tasks. As a baseline, we also evaluate the performance of triplet loss with negative data mining[4] (triplet-nm). In our experiments, we draw a pair of examples from two different classes and then form two triplets: each with one of the positive examples as query, the other one as positive, (any) one of the negative examples as negative. Thus, a batch of $2N$ training examples can produce $N = \frac{2N}{4} \times 2$ triplets, which is more efficient than the formulation in Equation (2) that we need $3N$ examples to form $N$ triplets. We adapt the smooth upper bound of triplet loss in Equation (4) instead of large-margin formulation [27] in all our experiments to be consistent with $N$-pair-mc losses.

We use Adam [11] for mini-batch stochastic gradient descent with data augmentation, namely horizontal flips and random crops. For evaluation, we extract a feature vector and compute the cosine similarity for verification. When more than one feature vectors are extracted via horizontal flip or from multiple crops, we use the cosine similarity averaged over all possible combinations between feature vectors of two examples. For all our experiments except for the face verification, we use ImageNet pretrained GoogLeNet[5] [23] for network initialization; for face verification, we use the same network architecture as CasiaNet [31] but trained from scratch without the last fully-connected layer for softmax classification. Our implementation is based on Caffe [10].

### 4.1   Fine-grained visual object recognition and verification

We evaluate deep metric learning algorithms on fine-grained object recognition and verification tasks. Specifically, we consider car and flower recognition problems on the following database:

- Car-333 [29] dataset is composed of $164,863$ images of cars from $333$ model categories collected from the internet. Following the experimental protocol [29], we split the dataset into $157,023$ images for training and $7,840$ for testing.

- Flower-610 dataset contains $61,771$ images of flowers from $610$ different flower species and among all collected, $58,721$ images are used for training and $3,050$ for testing.

We train networks for $40k$ iterations with $144$ examples per batch. This corresponds to 72 pairs per batch for $N$-pair losses. We perform 5-fold cross-validation on the training set and report the average performance on the test set. We evaluate both recognition and verification accuracy. Specifically, we consider verification setting where there are different number of negative examples from different classes, and determine as success only when the positive example is closer to the query example than any other negative example. Since the recognition task is involved, we also evaluate the performance of deep networks trained with softmax loss. The summary results are given in Table 1.

We observe consistent improvement of 72-pair loss models over triplet loss models. Although the negative data mining could bring substantial improvement to the baseline models, the performance is not as competitive as 72-pair loss models. Moreover, the 72-pair loss models are trained without negative data mining, thus should be more effective for deep metric learning framework. Between

| Database, evaluation metric | | triplet | triplet-nm | 72-pair-ovo | 72-pair-mc | softmax |
|---|---|---|---|---|---|---|
| Car-333 | Recognition | $70.24\pm0.38$ | $83.22\pm0.09$ | $86.84\pm0.13$ | $88.37\pm0.05$ | $\mathbf{89.21}\pm0.16$ $88.69\pm0.20^{\dagger}$ |
| | VRF (neg=1) | $96.78\pm0.04$ | $97.39\pm0.07$ | $\mathbf{98.09}\pm0.07$ | $97.92\pm0.06$ | $96.19\pm0.07$ |
| | VRF (neg=71) | $48.96\pm0.35$ | $65.14\pm0.24$ | $73.05\pm0.25$ | $\mathbf{76.02}\pm0.30$ | $55.36\pm0.30$ |
| Flower-610 | Recognition | $71.55\pm0.26$ | $82.85\pm0.22$ | $84.10\pm0.42$ | $\mathbf{85.57}\pm0.25$ | $84.38\pm0.28$ $84.59\pm0.21^{\dagger}$ |
| | VRF (neg=1) | $98.73\pm0.03$ | $99.15\pm0.03$ | $99.32\pm0.03$ | $\mathbf{99.50}\pm0.02$ | $98.72\pm0.04$ |
| | VRF (neg=71) | $73.04\pm0.13$ | $83.13\pm0.15$ | $87.42\pm0.18$ | $\mathbf{88.63}\pm0.14$ | $78.44\pm0.33$ |

Table 1: Mean recognition and verification accuracy with standard error on the test set of Car-333 and Flower-610 datasets. The recognition accuracy of all models are evaluated using $k$NN classifier; for models with softmax classifier, we also evaluate recognition accuracy using softmax classifier ($^{\dagger}$). The verification accuracy (VRF) is evaluated at different numbers of negative examples.

$N$-pair loss models, the multi-class loss (72-pair-mc) shows better performance than the one-vs-one loss (72-pair-ovo). As discussed in Section 3.1, superior performance of multi-class formulation is expected since the $N$-pair-ovo loss is decoupled in the sense that the individual losses are generated for each negative example independently.

When it compares to the softmax loss, the recognition performance of the 72-pair-mc loss models are competitive, showing slightly worse on Car-333 but better on Flower-610 datasets. However, the performance of softmax loss model breaks down severely on the verification task. We argue that the representation of the model trained with classification loss is not optimal for verification tasks. For example, examples near the classification decision boundary can still be classified correctly, but are prone to be missed for verification when there are examples from different class near the boundary.

## 4.2 Distance metric learning for unseen object recognition

Distance metric learning allows to learn a metric that can be generalized to an unseen categories. We highlight this aspect of deep metric learning on several visual object recognition benchmark. Following the experimental protocol in [21], we evaluate on the following three datasets:

- Stanford Online Product [21] dataset is composed of $120,053$ images from $22,634$ online product categories, and is partitioned into $59,551$ images of $11,318$ categories for training and $60,502$ images of $11,316$ categories for testing.

- Stanford Car-196 [12] dataset is composed of $16,185$ images of cars from 196 model categories. The first 98 model categories are used for training and the rest for testing.

- Caltech-UCSD Birds (CUB-200) [25] dataset is composed of $11,788$ images of birds from 200 different species. Similarly, we use the first 100 categories for training.

Unlike in Section 4.1, the object categories between train and test sets are disjoint. This makes the problem more challenging since deep networks can easily overfit to the categories in the train set and generalization of distance metric to unseen object categories could be difficult.

We closely follow experimental setting of [21]. For example, we initialize the network using ImageNet pretrained GoogLeNet and train for $20k$ iterations using the same network architecture (e.g., 64 dimensional embedding for Car-196 and CUB-200 datasets and 512 dimensional embedding for Online product dataset) and the number of examples (e.g., 120 examples) per batch. Besides, we use Adam for stochastic optimization and other hyperparameters such as learning rate are tuned accordingly via 5-fold cross-validation on the train set. We report the performance for both clustering and retrieval tasks using F1 and normalized mutual information (NMI) [16] scores for clustering as well as recall@$K$ [9] score for retrieval in Table 2.

We observe similar trend as in Section 4.1. The triplet loss model performs the worst among all losses considered. Negative data mining can alleviate the model to escape from the local optimum, but the $N$-pair loss models outperforms even without additional computational cost for negative data mining. The performance of $N$-pair loss further improves when combined with the proposed negative data mining. Overall, we improve by $9.6\%$ on F1 score, $1.99\%$ on NMI score, and $14.41\%$ on recall@1 score on Online product dataset compared to the baseline triplet loss models. Lastly, our model outperforms the performance of triplet loss with lifted structure [21], which demonstrates the effectiveness of the proposed $N$ pair batch construction.

| Online product | | | | | | | |
|---|---|---|---|---|---|---|---|
| | triplet | triplet-nm | triplet-lifted structure [21] | 60-pair-ovo | 60-pair-ovo-nm | 60-pair-mc | 60-pair-mc-nm |
| F1 | 19.59 | 24.27 | 25.6 | 23.13 | 25.31 | 26.53 | **28.19** |
| NMI | 86.11 | 87.23 | 87.5 | 86.98 | 87.45 | 87.77 | **88.10** |
| $K$=1 | 53.32 | 62.39 | 61.8 | 60.71 | 63.85 | 65.25 | **67.73** |
| $K$=10 | 72.75 | 79.69 | 79.9 | 78.74 | 81.22 | 82.15 | **83.76** |
| $K$=100 | 87.66 | 91.10 | 91.1 | 91.03 | 91.89 | 92.60 | **92.98** |
| $K$=1000 | 96.43 | 97.25 | 97.3 | 97.50 | 97.51 | 97.92 | **97.81** |

| | Car-196 | | | | CUB-200 | | | |
|---|---|---|---|---|---|---|---|---|
| | triplet | triplet-nm | 60-pair-ovo | 60-pair-mc | triplet | triplet-nm | 60-pair-ovo | 60-pair-mc |
| F1 | 24.73 | 27.86 | **33.52** | **33.55** | 21.88 | 24.37 | 25.21 | **27.24** |
| NMI | 58.25 | 59.94 | **63.87** | **63.95** | 55.83 | 57.87 | 58.55 | **60.39** |
| $K$=1 | 53.84 | 61.62 | 69.52 | **71.12** | 43.30 | 46.47 | 48.73 | **50.96** |
| $K$=2 | 66.02 | 73.48 | 78.76 | **79.74** | 55.84 | 58.58 | 60.48 | **63.34** |
| $K$=4 | 75.91 | 81.88 | 85.80 | **86.48** | 67.30 | 71.03 | 72.08 | **74.29** |
| $K$=8 | 84.18 | 87.81 | 90.94 | **91.60** | 77.48 | 80.17 | 81.62 | **83.22** |

Table 2: F1, NMI, and recall@$K$ scores on the test set of online product [21], Car-196 [12], and CUB-200 [25] datasets. F1 and NMI scores are averaged over 10 different random seeds for kmeans clustering but standard errors are omitted due to space limit. The best performing model and those with overlapping standard errors are bold-faced.

| | triplet | triplet-nm | 192-pair-ovo | 192-pair-mc | 320-pair-mc |
|---|---|---|---|---|---|
| VRF | 95.88±0.30 | 96.68±0.30 | 96.92±0.24 | **98.27**±0.19 | **98.33**±0.17 |
| Rank-1 | 55.14 | 60.93 | 66.21 | 88.58 | **90.17** |
| DIR@FIR=1% | 25.96 | 34.60 | 34.14 | 66.51 | **71.76** |

Table 3: Mean verification accuracy (VRF) with standard error, rank-1 accuracy of closed set identification and DIR@FAR=1% rate of open-set identification [1] on LFW dataset. The number of examples per batch is fixed to 384 for all models except for 320-pair-mc model.

## 4.3 Face verification and identification

Finally, we apply our deep metric learning algorithms on face verification and identification, a problem that determines whether two face images are the same identities (verification) and a problem that identifies the face image of the same identity from the gallery with many negative examples (identification). We train our networks on the WebFace database [31], which is composed of $494,414$ images from $10,575$ identities, and evaluate the quality of embedding networks trained with different metric learning objectives on Labeled Faces in the Wild (LFW) [8] database. We follow the network architecture in [31]. All networks are trained for $240k$ iterations, while the learning rate is decreased from $0.0003$ to $0.0001$ and $0.00003$ at $160k$ and $200k$ iterations, respectively. We report the performance of face verification. The summary result is provided in Table 3.

The triplet loss model shows $95.88\%$ verification accuracy, but the performance breaks down on identification tasks. Although negative data mining helps, the improvement is limited. Compared to these, the $N$-pair-mc loss model improves the performance by a significant margin. Furthermore, we observe additional improvement by increasing $N$ to 320, obtaining $98.33\%$ for verification, $90.17\%$ for closed-set and $71.76\%$ for open-set identification accuracy. It is worth noting that, although it shows better performance than the baseline triplet loss models, the $N$-pair-ovo loss model performs much worse than the $N$-pair-mc loss on this problem.

Interestingly, the $N$-pair-mc loss model also outperforms the model trained with combined contrastive loss and softmax loss whose verification accuracy is reported as $96.13\%$ [31]. Since this model is trained on the same dataset using the same network architecture, this clearly demonstrates the effectiveness of our proposed metric learning objectives on face recognition tasks. Nevertheless, there are other works reported higher accuracy for face verification. For example, [19] demonstrated $99.63\%$ test set verification accuracy on LFW database using triplet network trained with hundred millions of examples and [22] reported $98.97\%$ by training multiple deep neural networks from different facial keypoint regions with combined contrastive loss and softmax loss. Since our contribution is complementary to the scale of the training data or the network architecture, it is expected to bring further improvement by replacing the existing training objectives into our proposal.

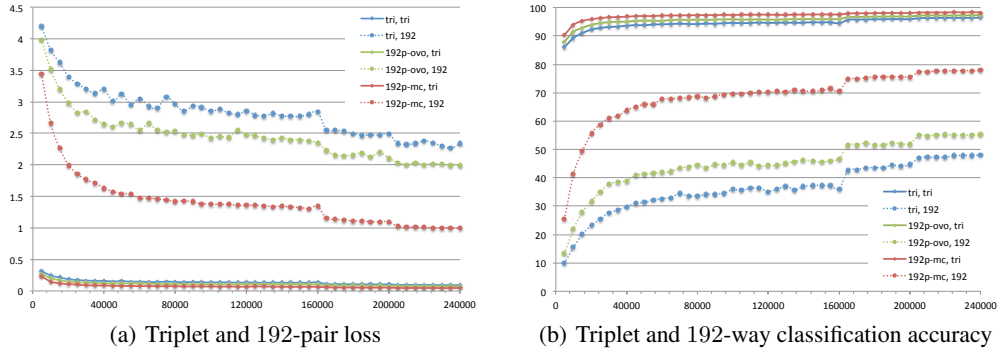

(a) Triplet and 192-pair loss         (b) Triplet and 192-way classification accuracy

Figure 3: Training curve of triplet, 192-pair-ovo, and 192-pair-mc loss models on WebFace database. We measure both (a) triplet and 192-pair loss as well as (b) classification accuracy.

| | Online product | | Car-196 | | | CUB-200 | | |
|---|---|---|---|---|---|---|---|---|
| | $60 \times 2$ | $30 \times 4$ | $60 \times 2$ | $30 \times 4$ | $10 \times 12$ | $60 \times 2$ | $30 \times 4$ | $10 \times 12$ |
| F1 | 26.53 | 25.01 | 33.55 | 31.92 | 29.87 | 27.24 | 27.54 | 26.66 |
| NMI | 87.77 | 87.40 | 63.87 | 62.94 | 61.84 | 60.39 | 60.43 | 59.37 |
| $K$=1 | 65.25 | 63.58 | 71.12 | 69.30 | 65.49 | 50.96 | 50.91 | 49.65 |

| | $192 \times 2$ | $96 \times 4$ | $64 \times 6$ | $32 \times 12$ |
|---|---|---|---|---|
| VRF | 98.27±0.19 | 98.25±0.25 | 97.98±0.22 | 97.57±0.33 |
| Rank-1 | 88.58 | 87.53 | 83.96 | 79.61 |
| DIR@FIR=1% | 66.51 | 66.22 | 64.38 | 56.46 |

Table 4: F1, NMI, and recall@1 scores on online product, Car-196, and CUB-200 datasets, and verification and rank-1 accuracy on LFW database. For model name of $N \times M$, we refer $N$ the number of different classes in each batch and $M$ the number of positive examples per class.

Finally, we provide training curve in Figure 3. Since the difference of triplet loss between models is relatively small, we also measure 192-pair loss (and accuracy) of three models at every $5k$ iteration. We observe significantly faster training progress using 192-pair-mc loss than triplet loss; it only takes $15k$ iterations to reach at the loss at convergence of triplet loss model ($240k$ iteration).

### 4.4 Analysis on tuplet construction methods

In this section, we highlight the importance of the proposed tuplet construction strategy using $N$ pairs of examples by conducting control experiments using different numbers of distinguishable classes per batch while fixing the total number of examples per batch the same. For example, if we are to use $N/2$ different classes per batch rather than $N$ different classes, we select 4 examples from each class instead of a pair of examples. Since $N$-pair loss is not defined to handle multiple positive examples, we follow the definition of NCA in this experiments as follows:

$$\mathcal{L} = \frac{1}{2N} \sum_i - \log \frac{\sum_{j \neq i : y_j = y_i} \exp(f_i^\top f_j)}{\sum_{j \neq i} \exp(f_i^\top f_j)} \qquad (9)$$

We repeat experiments in Section 4.2 and 4.3 and provide the summary results in Table 4. We observe a certain degree of performance drop as we decrease the number of classes. Despite, all of these results are substantially better than those of triplet loss, confirming the importance of training with multiple negative classes, and suggesting to train with as many negative classes as possible.

## 5 Conclusion

Triplet loss has been widely used for deep metric learning, even though with somewhat unsatisfactory convergence. We present a scalable novel objective, multi-calss $N$-pair loss, for deep metric learning, which significantly improves upon the triplet loss by pushing away multiple negative examples jointly at each update. We demonstrate the effectiveness of $N$-pair-mc loss on fine-grained visual recognition and verification, as well as visual object clustering and retrieval.

### Acknowledgments

We express our sincere thanks to Wenling Shang for her support in many parts of this work from algorithm development to paper writing. We also thank Junhyuk Oh and Paul Vernaza for helpful discussion.

## Footnotes

[1]We assume $f$ to have unit norm in Equation (5) to avoid degeneracy.

[2]We also consider the symmetric loss to Equation (7) that swaps $f$ and $f^+$ to maximize the efficacy.

[3]Our $N$-pair batch construction can be seen as a special case of lifted structure [21] where the batch includes only positive pairs that are from disjoint classes. Besides, the loss function in [21] is based on the max-margin formulation, whereas we optimize the log probability of identification loss directly.

[4]Throughout experiments, negative data mining refers to the negative class mining for both triplet and $N$-pair loss instead of negative instance mining.

[5]https://github.com/BVLC/caffe/tree/master/models/bvlc_googlenet

# References

[1] L. Best-Rowden, H. Han, C. Otto, B. F. Klare, and A. K. Jain. Unconstrained face recognition: Identifying a person of interest from a media collection. *IEEE Transactions on Information Forensics and Security*, 9(12):2144–2157, 2014.

[2] G. Chechik, V. Sharma, U. Shalit, and S. Bengio. Large scale online learning of image similarity through ranking. *Journal of Machine Learning Research*, 11:1109–1135, 2010.

[3] S. Chopra, R. Hadsell, and Y. LeCun. Learning a similarity metric discriminatively, with application to face verification. In *CVPR*, 2005.

[4] Y. Cui, F. Zhou, Y. Lin, and S. Belongie. Fine-grained categorization and dataset bootstrapping using deep metric learning with humans in the loop. In *CVPR*, 2016.

[5] R. Girshick, J. Donahue, T. Darrell, and J. Malik. Region-based convolutional networks for accurate object detection and segmentation. *IEEE Transactions on Pattern Analysis and Machine Intelligence*, PP(99):1–1, 2015.

[6] J. Goldberger, G. E. Hinton, S. T. Roweis, and R. Salakhutdinov. Neighbourhood components analysis. In *NIPS*, 2004.

[7] R. Hadsell, S. Chopra, and Y. LeCun. Dimensionality reduction by learning an invariant mapping. In *CVPR*, 2006.

[8] G. B. Huang, M. Narayana, and E. Learned-Miller. Towards unconstrained face recognition. In *CVPR Workshop*, 2008.

[9] H. Jegou, M. Douze, and C. Schmid. Product quantization for nearest neighbor search. *IEEE Transactions on Pattern Analysis and Machine Intelligence*, 33(1):117–128, 2011.

[10] Y. Jia, E. Shelhamer, J. Donahue, S. Karayev, J. Long, R. Girshick, S. Guadarrama, and T. Darrell. Caffe: Convolutional architecture for fast feature embedding. *arXiv preprint arXiv:1408.5093*, 2014.

[11] D. Kingma and J. Ba. Adam: A method for stochastic optimization. In *ICLR*, 2015.

[12] J. Krause, M. Stark, J. Deng, and L. Fei-Fei. 3d object representations for fine-grained categorization. In *ICCV Workshop*, 2013.

[13] A. Krizhevsky, I. Sutskever, and G. E. Hinton. ImageNet classification with deep convolutional neural networks. In *NIPS*, 2012.

[14] J. Liu, Y. Deng, T. Bai, and C. Huang. Targeting ultimate accuracy: Face recognition via deep embedding. *CoRR*, abs/1506.07310, 2015.

[15] D. G. Lowe. Similarity metric learning for a variable-kernel classifier. *Neural computation*, 7(1):72–85, 1995.

[16] C. D. Manning, P. Raghavan, H. Schütze, et al. *Introduction to information retrieval*, volume 1. Cambridge university press Cambridge, 2008.

[17] M. Norouzi, D. J. Fleet, and R. R. Salakhutdinov. Hamming distance metric learning. In *NIPS*, 2012.

[18] O. M. Parkhi, A. Vedaldi, and A. Zisserman. Deep face recognition. *BMVC*, 2015.

[19] F. Schroff, D. Kalenichenko, and J. Philbin. FaceNet: A unified embedding for face recognition and clustering. In *CVPR*, 2015.

[20] K. Simonyan and A. Zisserman. Very deep convolutional networks for large-scale image recognition. In *ICLR*, 2015.

[21] H. O. Song, Y. Xiang, S. Jegelka, and S. Savarese. Deep metric learning via lifted structured feature embedding. In *CVPR*, 2016.

[22] Y. Sun, Y. Chen, X. Wang, and X. Tang. Deep learning face representation by joint identification-verification. In *NIPS*, 2014.

[23] C. Szegedy, W. Liu, Y. Jia, P. Sermanet, S. Reed, D. Anguelov, D. Erhan, V. Vanhoucke, and A. Rabinovich. Going deeper with convolutions. In *CVPR*, 2015.

[24] Y. Taigman, M. Yang, M. Ranzato, and L. Wolf. Deepface: Closing the gap to human-level performance in face verification. In *CVPR*, 2014.

[25] C. Wah, S. Branson, P. Welinder, P. Perona, and S. Belongie. The Caltech-UCSD Birds-200-2011 Dataset. Technical Report CNS-TR-2011-001, California Institute of Technology, 2011.

[26] J. Wang, Y. Song, T. Leung, C. Rosenberg, J. Wang, J. Philbin, B. Chen, and Y. Wu. Learning fine-grained image similarity with deep ranking. In *CVPR*, 2014.

[27] K. Q. Weinberger, J. Blitzer, and L. K. Saul. Distance metric learning for large margin nearest neighbor classification. In *NIPS*, 2005.

[28] J. Weston, S. Bengio, and N. Usunier. Wsabie: Scaling up to large vocabulary image annotation. In *IJCAI*, volume 11, pages 2764–2770, 2011.

[29] S. Xie, T. Yang, X. Wang, and Y. Lin. Hyper-class augmented and regularized deep learning for fine-grained image classification. In *CVPR*, 2015.

[30] E. P. Xing, A. Y. Ng, M. I. Jordan, and S. Russell. Distance metric learning with application to clustering with side-information. 2003.

[31] D. Yi, Z. Lei, S. Liao, and S. Z. Li. Learning face representation from scratch. *CoRR*, abs/1411.7923, 2014.

[32] X. Zhang, F. Zhou, Y. Lin, and S. Zhang. Embedding label structures for fine-grained feature representation. In *CVPR*, 2016.

